# Privacy-preserving logistic regression

**Kamalika Chaudhuri**
Information Theory and Applications
University of California, San Diego
kamalika@soe.ucsd.edu

**Claire Monteleoni**[*]
Center for Computational Learning Systems
Columbia University
cmontel@ccls.columbia.edu

## Abstract

This paper addresses the important tradeoff between privacy and learnability, when designing algorithms for learning from private databases. We focus on privacy-preserving logistic regression. First we apply an idea of Dwork *et al.* [6] to design a privacy-preserving logistic regression algorithm. This involves bounding the sensitivity of regularized logistic regression, and perturbing the learned classifier with noise proportional to the sensitivity.

We then provide a privacy-preserving regularized logistic regression algorithm based on a new privacy-preserving technique: solving a perturbed optimization problem. We prove that our algorithm preserves privacy in the model due to [6]. We provide learning guarantees for both algorithms, which are tighter for our new algorithm, in cases in which one would typically apply logistic regression. Experiments demonstrate improved learning performance of our method, versus the sensitivity method. Our privacy-preserving technique does not depend on the sensitivity of the function, and extends easily to a class of convex loss functions. Our work also reveals an interesting connection between regularization and privacy.

## 1 Introduction

Privacy-preserving machine learning is an emerging problem, due in part to the increased reliance on the internet for day-to-day tasks such as banking, shopping, and social networking. Moreover, private data such as medical and financial records are increasingly being digitized, stored, and managed by independent companies. In the literature on cryptography and information security, data privacy definitions have been proposed, however designing machine learning algorithms that adhere to them has not been well-explored. On the other hand, data-mining algorithms have been introduced that aim to respect other notions of privacy that may be less formally justified.

Our goal is to bridge the gap between approaches in the cryptography and information security community, and those in the data-mining community. This is necessary, as there is a tradeoff between the privacy of a protocol, and the learnability of functions that respect the protocol. In addition to the specific contributions of our paper, we hope to encourage the machine learning community to embrace the goals of privacy-preserving machine learning, as it is still a fledgling endeavor.

In this work, we provide algorithms for learning in a privacy model introduced by Dwork *et al.* [6]. The $\epsilon$-*differential privacy* model limits how much information an adversary can gain about a particular private value, by observing a function learned from a database containing that value, even if she knows every other value in the database. An initial positive result [6] in this setting depends on the *sensitivity* of the function to be learned, which is the maximum amount the function value can change due to an arbitrary change in one input. Using this method requires bounding the sensitivity of the function class to be learned, and then adding noise proportional to the sensitivity. This might be difficult for some functions that are important for machine learning.

---

[*]The majority of this work was done while at UC San Diego.

The contributions of this paper are as follows. First we apply the sensitivity-based method of designing privacy-preserving algorithms [6] to a specific machine learning algorithm, logistic regression. Then we present a second privacy-preserving logistic regression algorithm. The second algorithm is based on solving a perturbed objective function, and does not depend on the sensitivity. We prove that the new method is private in the $\epsilon$-differential privacy model. We provide learning performance guarantees for both algorithms, which are tighter for our new algorithm, in cases in which one would typically apply logistic regression. Finally, we provide experiments demonstrating superior learning performance of our new method, with respect to the algorithm based on [6]. Our technique may have broader applications, and we show that it can be applied to certain classes of optimization problems.

## 1.1 Overview and related work

At the first glance, it may seem that anonymizing a data-set – namely, stripping it of identifying information about individuals, such as names, addresses, etc – is sufficient to preserve privacy. However, this is problematic, because an adversary may have some auxiliary information, which may even be publicly available, and which can be used to breach privacy. For more details on such attacks, see [12].

To formally address this issue, we need a definition of privacy which works in the presence of auxiliary knowledge by the adversary. The definition we use is due to Dwork *et al.* [6], and has been used in several applications [4, 11, 2]. We explain this definition and privacy model in more detail in Section 2.

**Privacy and learning.** The work most related to ours is [8] and [3]. [8] shows how to find classifiers that preserve $\epsilon$-differential privacy; however, their algorithm takes time exponential in $d$ for inputs in $\mathbf{R}^d$. [3] provides a general method for publishing data-sets while preserving $\epsilon$-differential privacy such that the answers to all queries of a certain type with low VC-dimension are approximately correct. However, their algorithm can also be computationally inefficient.

**Additional related work.** There has been a substantial amount of work on privacy in the literature, spanning several communities. Much work on privacy has been done in the data-mining community [1, 7], [14, 10], however the privacy definitions used in these papers are different, and some are susceptible to attacks when the adversary has some prior information. In contrast, the privacy definition we use avoids these attacks, and is very strong.

## 2 Sensitivity and the $\epsilon$-differential privacy model

Before we define the privacy model that we study, we will note a few preliminary points. Both in that model, and for our algorithm and analyses, we assume that each value in the database is a real vector with norm at most one. That is, a database contains values $x_1, \ldots, x_n$, where $x_i \in \mathbf{R}^d$, and $\|x_i\| \leq 1$ for all $i \in \{1, \ldots, n\}$. This assumption is used in the privacy model. In addition, we assume that when learning linear separators, the best separator passes through the origin. Note that this is not an assumption that the data is separable, but instead an assumption that a vector's classification is based on its angle, regardless of its norm.

In both privacy-preserving logistic regression algorithms that we state, the output is a parameter vector, $w$, which makes prediction $\text{SGN}(w \cdot x)$, on a point $x$. For a vector $x$, we use $\|x\|$ to denote its Euclidean norm. For a function $G(x)$ defined on $\mathbf{R}^d$, we use $\nabla G$ to denote its gradient and $\nabla^2 G$ to denote its Hessian.

**Privacy Definition.** The privacy definition we use is due to Dwork *et al.* [6, 5]. In this model, users have access to private data about individuals through a *sanitization mechanism*, usually denoted by $M$. The interaction between the sanitization mechanism and an adversary is modelled as a sequence of queries, made by the adversary, and responses, made by the sanitizer. The sanitizer, which is typically a randomized algorithm, is said to preserve $\epsilon$-differential privacy, if the private value of any one individual in the data set does not affect the likelihood of a specific answer by the sanitizer by more than $\epsilon$.

More formally, $\epsilon$-differential privacy can be defined as follows.

**Definition 1** *A randomized mechanism $M$ provides $\epsilon$-differential privacy, if, for all databases $D_1$ and $D_2$ which differ by at most one element, and for any $t$,*

$$\frac{\Pr[M(D_1) = t]}{\Pr[M(D_2) = t]} \leq e^\epsilon$$

It was shown in [6] that if a mechanism satisfies $\epsilon$-differential privacy, then an adversary who knows the private value of all the individuals in the data-set, except for one single individual, cannot figure out the private value of the unknown individual, with sufficient confidence, from the responses of the sanitizer. $\epsilon$-differential privacy is therefore a very strong notion of privacy.

[6] also provides a general method for computing an approximation to any function $f$ while preserving $\epsilon$-differential privacy. Before we can describe their method, we need a definition.

**Definition 2** *For any function $f$ with $n$ inputs, we define the sensitivity $S(f)$ as the maximum, over all inputs, of the difference in the value of $f$ when one input of $f$ is changed. That is,*

$$S(f) = \max_{(a,a')} |f(x_1, \ldots, x_{n-1}, x_n = a) - f(x_1, \ldots, x_{n-1}, x_n = a')|$$

[6] shows that for any input $x_1, \ldots, x_n$, releasing $f(x_1, \ldots, x_n) + \eta$, where $\eta$ is a random variable drawn from a Laplace distribution with mean 0 and standard deviation $\frac{S(f)}{\epsilon}$ preserves $\epsilon$-differential privacy.

In [13], Nissim *et al.* showed that given any input $x$ to a function, and a function $f$, it is sufficient to draw $\eta$ from a Laplace distribution with standard deviation $\frac{SS(f)}{\epsilon}$, where $SS(f)$ is the *smoothed-sensitivity* of $f$ around $x$. Although this method sometimes requires adding a smaller amount of noise to preserve privacy, in general, smoothed sensitivity of a function can be hard to compute.

## 3 A Simple Algorithm

Based on [6], one can come up with a simple algorithm for privacy-preserving logistic regression, which adds noise to the classifier obtained by logistic regression, proportional to its sensitivity. From Corollary 2, the sensitivity of logistic regression is at most $\frac{2}{n\lambda}$. This leads to Algorithm 1, which obeys the privacy guarantees in Theorem 1.

**Algorithm 1:**

1. Compute $w^*$, the classifier obtained by regularized logistic regression on the labelled examples $(x_1, y_1), \ldots, (x_n, y_n)$.

2. Pick a noise vector $\eta$ according to the following density function: $h(\eta) \propto e^{-\frac{n\epsilon\lambda}{2}||\eta||}$. To pick such a vector, we choose the norm of $\eta$ from the $\Gamma(d, \frac{2}{n\epsilon\lambda})$ distribution, and the direction of $\eta$ uniformly at random.

3. Output $w^* + \eta$.

**Theorem 1** *Let $(x_1, y_1), \ldots, (x_n, y_n)$ be a set of labelled points over $\mathbf{R}^d$ such that $||x_i|| \leq 1$ for all $i$. Then, Algorithm 1 preserves $\epsilon$-differential privacy.*

PROOF: The proof follows by a combination of [6], and Corollary 2, which states that the sensitivity of logistic regression is at most $\frac{2}{n\lambda}$. □

**Lemma 1** *Let $G(w)$ and $g(w)$ be two convex functions, which are continuous and differentiable at all points. If $w_1 = \operatorname{argmin}_w G(w)$ and $w_2 = \operatorname{argmin}_w G(w) + g(w)$, then, $||w_1 - w_2|| \leq \frac{g_1}{G_2}$. Here, $g_1 = \max_w ||\nabla g(w)||$ and $G_2 = \min_v \min_w v^T \nabla^2 G(w)v$, for any unit vector $v$.*

The main idea of the proof is to examine the gradient and the Hessian of the functions $G$ and $g$ around $w_1$ and $w_2$. Due to lack of space, the full proof appears in the full version of our paper.

**Corollary 2** *Given a set of $n$ examples $x_1, \ldots, x_n$ in $\mathbf{R}^d$, with labels $y_1, \ldots, y_n$, such that for all $i$, $||x_i|| \leq 1$, the sensitivity of logistic regression with regularization parameter $\lambda$ is at most $\frac{2}{n\lambda}$.*

PROOF: We use a triangle inequality and the fact that $G_2 \geq \lambda$ and $g_1 \leq \frac{1}{n}$. $\square$

**Learning Performance.** In order to assess the performance of Algorithm 1, we first try to bound the performance of Algorithm 1 on the training data. To do this, we need to define some notation.

For a classifier $w$, we use $L(w)$ to denote the expected loss of $w$ over the data distribution, and $\hat{L}(w)$ to denote the empirical average loss of $w$ over the training data. In other words, $\hat{L}(w) = \frac{1}{n} \sum_{i=1}^{n} \log(1 + e^{-y_i w^T x_i})$, where, $(x_i, y_i), i = 1, \ldots, n$ are the training examples.

Further, for a classifier $w$, we use the notation $f_\lambda(w)$ to denote the quantity $\frac{1}{2}\lambda ||w||^2 + L(w)$ and $\hat{f}_\lambda(w)$ to denote the quantity $\frac{1}{2}\lambda ||w||^2 + \hat{L}(w)$. Our guarantees on this algorithm can be summarized by the following lemma.

**Lemma 3** *Given a logistic regression problem with regularization parameter $\lambda$, let $w_1$ be the classifier that minimizes $\hat{f}_\lambda$, and $w_2$ be the classifier output by Algorithm 1 respectively. Then, with probability $1 - \delta$ over the randomness in the privacy mechanism, $\hat{f}_\lambda(w_2) \leq \hat{f}_\lambda(w_1) + \frac{2d^2(1+\lambda)\log^2(d/\delta)}{\lambda^2 n^2 \epsilon^2}$.*

Due to lack of space, the proof is deferred to the full version.

From Lemma 3, we see that performance of Algorithm 1 degrades with decreasing $\lambda$, and is poor in particular when $\lambda$ is very small. One question is, can we get a privacy-preserving approximation to logistic regression, which has better performance bounds for small $\lambda$? To explore this, in the next section, we look at a different algorithm.

# 4 A New Algorithm

In this section, we provide a new privacy-preserving algorithm for logistic regression. The input to our algorithm is a set of examples $x_1, \ldots, x_n$ over $\mathbf{R}^d$ such that $||x_i|| \leq 1$ for all $i$, a set of labels $y_1, \ldots, y_n$ for the examples, a regularization constant $\lambda$ and a privacy parameter $\epsilon$, and the output is a vector $w^*$ in $\mathbf{R}^d$. Our algorithm works as follows.

**Algorithm 2:**

1. We pick a random vector $b$ from the density function $h(b) \propto e^{-\frac{\epsilon}{2}||b||}$. To implement this, we pick the norm of $b$ from the $\Gamma(d, \frac{2}{\epsilon})$ distribution, and the direction of $b$ uniformly at random.

2. Given examples $x_1, \ldots, x_n$, with labels $y_1, \ldots, y_n$ and a regularization constant $\lambda$, we compute $w^* = \text{argmin}_w \frac{1}{2}\lambda w^T w + \frac{b^T w}{n} + \frac{1}{n}\sum_{i=1}^{n}\log(1 + e^{-y_i w^T x_i})$. Output $w^*$.

We observe that our method solves a convex programming problem very similar to the logistic regression convex program, and therefore it has running time similar to that of logistic regression. In the sequel, we show that the output of Algorithm 2 is privacy preserving.

**Theorem 2** *Given a set of $n$ examples $x_1, \ldots, x_n$ over $\mathbf{R}^d$, with labels $y_1, \ldots, y_n$, where for each $i$, $||x_i|| \leq 1$, the output of Algorithm 2 preserves $\epsilon$-differential privacy.*

PROOF: Let $a$ and $a'$ be any two vectors over $\mathbf{R}^d$ with norm at most 1, and $y, y' \in \{-1, 1\}$. For any such $(a, y), (a', y')$, consider the inputs $(x_1, y_1), \ldots, (x_{n-1}, y_{n-1}), (a, y)$ and $(x_1, y_1) \ldots, (x_{n-1}, y_{n-1}), (a', y')$. Then, for any $w^*$ output by our algorithm, there is a *unique* value of $b$ that maps the input to the output. This uniqueness holds, because both the regularization function and the loss functions are differentiable everywhere.

Let the values of $b$ for the first and second cases respectively, be $b_1$ and $b_2$.

Since $w^*$ is the value that minimizes both the optimization problems, the derivative of both optimization functions at $w^*$ is 0.

This implies that for every $b_1$ in the first case, there exists a $b_2$ in the second case such that: $b_1 - \frac{ya}{1 + e^{yw^{*T}a}} = b_2 - \frac{y'a'}{1 + e^{y'w^{*T}a'}}$. Since $||a|| \leq 1$, $||a'|| \leq 1$, and $\frac{1}{1 + e^{yw^{*T}a}} \leq 1$, and $\frac{1}{1 + e^{y'w^{*T}a'}} \leq 1$

for any $w^*$, $||b_1 - b_2|| \leq 2$. Using the triangle inequality, $||b_1|| - 2 \leq ||b_2|| \leq ||b_1|| + 2$. Therefore, for any pair $(a, y), (a', y')$,

$$\frac{\Pr[w^*|x_1, \ldots, x_{n-1}, y_1, \ldots, y_{n-1}, x_n = a, y_n = y]}{\Pr[w^*|x_1, \ldots, x_{n-1}, y_1, \ldots, y_{n-1}, x_n = a', y_n = y']} = \frac{h(b_1)}{h(b_2)} = e^{-\frac{\epsilon}{2}(||b_1|| - ||b_2||)}$$

where $h(b_i)$ for $i = 1, 2$ is the density of $b_i$. Since $-2 \leq ||b_1|| - ||b_2|| \leq 2$, this ratio is at most $e^\epsilon$. theorem follows. $\square$

We notice that the privacy guarantee for our algorithm does not depend on $\lambda$; in other words, for any value of $\lambda$, our algorithm is private. On the other hand, as we show in Section 5, the performance of our algorithm does degrade with decreasing $\lambda$ in the worst case, although the degradation is better than that of Algorithm 1 for $\lambda < 1$.

**Other Applications.** Our algorithm for privacy-preserving logistic regression can be generalized to provide privacy-preserving outputs for more general convex optimization problems, so long as the problems satisfy certain conditions. These conditions can be formalized in the theorem below.

**Theorem 3** *Let $X = \{x_1, \ldots, x_n\}$ be a database containing private data of individuals. Suppose we would like to compute a vector $w$ that minimizes the function $F(w) = G(w) + \sum_{i=1}^{n} l(w, x_i)$, over $w \in R^d$ for some $d$, such that all of the following hold:*

1. *$G(w)$ and $l(w, x_i)$ are differentiable everywhere, and have continuous derivatives*

2. *$G(w)$ is strongly convex and $l(w, x_i)$ are convex for all $i$*

3. *$||\nabla_w l(w, x)|| \leq \kappa$, for any $x$.*

*Let $b = B \cdot \hat{b}$, where $B$ is drawn from $\Gamma(d, \frac{2\kappa}{\epsilon})$, and $\hat{b}$ is drawn uniformly from the surface of a $d$-dimensional unit sphere. Then, computing $w^*$, where $w^* = \operatorname{argmin}_w G(w) + \sum_{i=1}^{n} l(w, x_i) + b^T w$, provides $\epsilon$-differential privacy.*

# 5 Learning Guarantees

In this section, we show theoretical bounds on the number of samples required by the algorithms to learn a linear classifier. For the rest of the section, we use the same notation used in Section 3.

First we show that, for Algorithm 2, the values of $\hat{f}_\lambda(w_2)$ and $\hat{f}_\lambda(w_1)$ are close.

**Lemma 4** *Given a logistic regression problem with regularization parameter $\lambda$, let $w_1$ be the classifier that minimizes $\hat{f}_\lambda$, and $w_2$ be the classifier output by Algorithm 2 respectively. Then, with probability $1 - \delta$ over the randomness in the privacy mechanism, $\hat{f}_\lambda(w_2) \leq \hat{f}_\lambda(w_1) + \frac{8d^2 \log^2(d/\delta)}{\lambda n^2 \epsilon^2}$.*

The proof is in the full version of our paper. As desired, for $\lambda < 1$, we have attained a tighter bound using Algorithm 2, than Lemma 3 for Algorithm 1.

Now we give a performance guarantee for Algorithm 2.

**Theorem 4** *Let $w_0$ be a classifier with expected loss $L$ over the data distribution. If the training examples are drawn independently from the data distribution, and if $n > C \max(\frac{||w_0||^2}{\epsilon_g^2}, \frac{d \log(\frac{d}{\delta}) ||w_0||}{\epsilon_g \epsilon})$, for some constant $C$, then, with probability $1 - \delta$, the classifier output by Algorithm 2 has loss at most $L + \epsilon_g$ over the data distribution.*

PROOF: Let $w^*$ be the classifier that minimizes $f_\lambda(w)$ over the data distribution, and let $w_1$ and $w_2$ be the classifiers that minimize $\hat{f}_\lambda(w)$ and $\hat{f}_\lambda(w) + \frac{b^T w}{n}$ over the data distribution respectively. We can use an analysis as in [15] to write that:

$$L(w_2) = L(w_0) + (f_\lambda(w_2) - f_\lambda(w^*)) + (f_\lambda(w^*) - f_\lambda(w_0)) + \frac{\lambda}{2}||w_0||^2 - \frac{\lambda}{2}||w_2||^2 \quad (1)$$

Notice that from Lemma 4, $\hat{f}_\lambda(w_2) - \hat{f}_\lambda(w_1) \leq \frac{8d^2 \log^2(d/\delta)}{\lambda n^2 \epsilon^2}$. Using this and [16], we can bound the second quantity in equation 1 as $f_\lambda(w_2) - f_\lambda(w^*) \leq \frac{16d^2 \log^2(d/\delta)}{\lambda n^2 \epsilon^2} + O(\frac{1}{\lambda n})$. By definition of $w^*$, the third quantity in Equation 1 is non-positive. If $\lambda$ is set to be $\frac{\epsilon_g}{||w_0||^2}$, then, the fourth quantity in Equation 1 is at most $\frac{\epsilon_g}{2}$. Now, if $n > C \cdot \frac{||w_0||^2}{\epsilon_g^2}$ for a suitable constant $C$, $\frac{1}{\lambda n} \leq \frac{\epsilon_g}{4}$. In addition, if $n > C \cdot \frac{||w_0|| d \log(\frac{d}{\delta})}{\epsilon \epsilon_g}$, then, $\frac{16d^2 \log^2(\frac{d}{\delta})}{\lambda n^2 \epsilon^2} \leq \frac{\epsilon_g}{4}$. In either case, the total loss of the classifier $w_2$ output by Algorithm 2 is at most $L(w_0) + \epsilon_g$.

$\square$

The same technique can be used to analyze the sensitivity-based algorithm, using Lemma 3, which yields the following.

**Theorem 5** *Let $w_0$ be a classifier with expected loss $L$ over the data distribution. If the training examples are drawn independently from the data distribution, and if $n > C \max(\frac{||w_0||^2}{\epsilon_g^2}, \frac{d \log(\frac{d}{\delta})||w_0||}{\epsilon_g \epsilon}, \frac{d \log(\frac{d}{\delta})||w_0||^2}{\epsilon_g^{3/2} \epsilon})$, for some constant $C$, then, with probability $1 - \delta$, the classifier output by Algorithm 2 has loss at most $L + \epsilon_g$ over the data distribution.*

It is clear that this bound is never lower than the bound for Algorithm 2. Note that for problems in which (non-private) logistic regression performs well, $\|w_0\| \geq 1$ if $w_0$ has low loss, since otherwise one can show that the loss of $w_0$ would be lower bounded by $\log(1 + \frac{1}{e})$. Thus the performance guarantee for Algorithm 2 is significantly stronger than for Algorithm 1, for problems in which one would typically apply logistic regression.

## 6   Results in Simulation

|  | Uniform, margin=0.03 | Unseparable (uniform with noise 0.2 in margin 0.1) |
|---|---|---|
| Sensitivity method | $0.2962_{\pm 0.0617}$ | $0.3257_{\pm 0.0536}$ |
| New method | $0.1426_{\pm 0.1284}$ | $0.1903_{\pm 0.1105}$ |
| Standard LR | $0_{\pm 0.0016}$ | $0.0530_{\pm 0.1105}$ |

Figure 1: Test error: mean $\pm$ standard deviation over five folds. N=17,500.

We include some simulations that compare the two privacy-preserving methods, and demonstrate that using our privacy-preserving approach to logistic regression does not degrade learning performance terribly, from that of standard logistic regression. Performance degradation is inevitable however, as in both cases, in order to address privacy concerns, we are adding noise, either to the learned classifier, or to the objective.

In order to obtain a clean comparison between the various logistic regression variants, we first experimented with artificial data that is separable through the origin. Because the classification of a vector by a linear separator through the origin depends only its angle, not its norm, we sampled the data from the unit hypersphere. We used a uniform distribution on the hypersphere in 10 dimensions with zero mass within a small margin (0.03) from the generating linear separator. Then we experimented on uniform data that is not linearly separable. We sampled data from the surface of the unit ball in 10 dimensions, and labeled it with a classifier through the origin. In the band of margin $\leq 0.1$ with respect to the perfect classifier, we performed random label flipping with probability 0.2. For our experiments, we used convex optimization software provided by [9].

Figure 1 gives mean and standard deviation of test error over five-fold cross-validation, on 17,500 points. In both simulations, our new method is superior to the sensitivity method, although incurs more error than standard (non-private) logistic regression. For both problems, we tuned the logistic regression parameter, $\lambda$, to minimize the test error of standard logistic regression, using five-fold cross-validation on a holdout set of 10,000 points (the tuned values are: $\lambda = 0.01$ in both cases). For each test error computation, the performance of each of the privacy-preserving algorithms was evaluated by averaging over 200 random restarts, since they are both randomized algorithms.

In Figure 2a)-b) we provide learning curves. We graph the test error after each increment of 1000 points, averaged over five-fold cross validation. The learning curves reveal that, not only does the

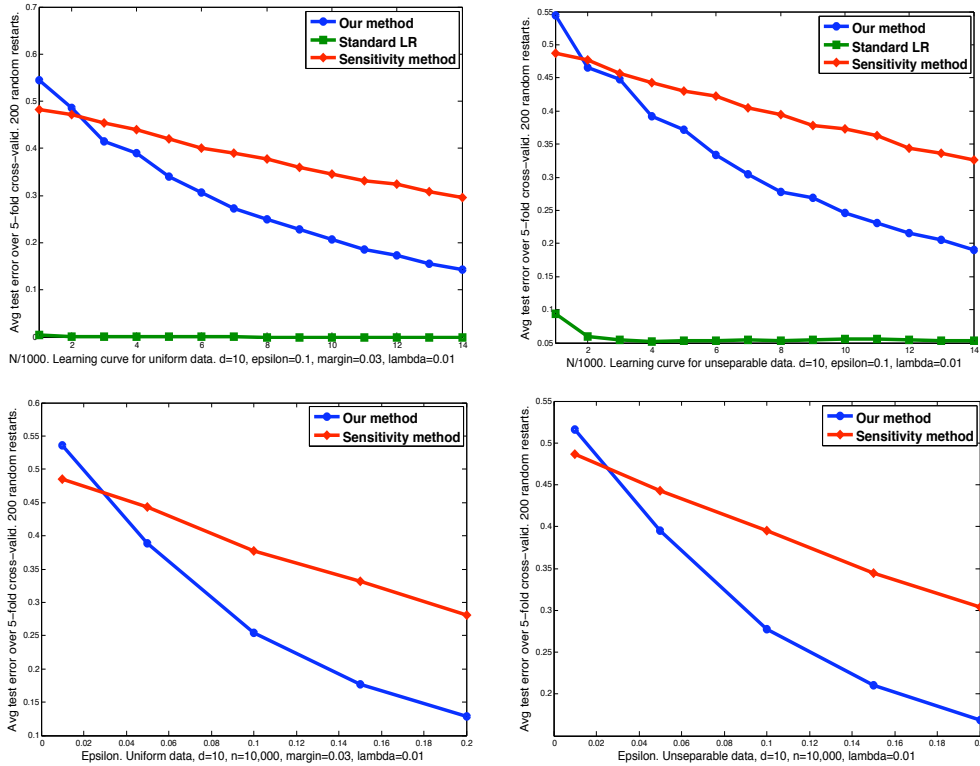

Figure 2: Learning curves: a) Uniform distribution, margin=0.03, b) Unseparable data. Epsilon curves: c) Uniform distribution, margin=0.03, d) Unseparable data.

new method reach a lower final error than the sensitivity method, but it also has better performance at most smaller training set sizes.

In order to observe the effect of the level of privacy on the learning performance of the privacy-preserving learning algorithms, in Figure 2c)-d) we vary $\epsilon$, the privacy parameter to the two algorithms, on both the uniform, low margin data, and the unseparable data. As per the definition of $\epsilon$-differential privacy in Section 2, strengthening the privacy guarantee corresponds to reducing $\epsilon$. Both algorithms' learning performance degrades in this direction. For the majority of values of $\epsilon$ that we tested, the new method is superior in managing the tradeoff between privacy and learning performance. For very small $\epsilon$, corresponding to extremely stringent privacy requirements, the sensitivity method performs better but also has a predication accuracy close to chance, which is not useful for machine learning purposes.

## 7   Conclusion

In conclusion, we show two ways to construct a privacy-preserving linear classifier through logistic regression. The first one uses the methods of [6], and the second one is a new algorithm. Using the $\epsilon$-differential privacy definition of Dwork *et al.* [6], we prove that our new algorithm is privacy-preserving. We provide learning performance guarantees for the two algorithms, which are tighter for our new algorithm, in cases in which one would typically apply logistic regression. In simulations, our new algorithm outperforms the method based on [6].

Our work reveals an interesting connection between regularization and privacy: the larger the regularization constant, the less sensitive the logistic regression function is to any one individual example, and as a result, the less noise one needs to add to make it privacy-preserving. Therefore, regularization not only prevents overfitting, but also helps with privacy, by making the classifier less

sensitive. An interesting future direction would be to explore whether other methods that prevent overfitting also have such properties.

Other future directions would be to apply our techniques to other commonly used machine-learning algorithms, and to explore whether our techniques can be applied to more general optimization problems. Theorem 3 shows that our method can be applied to a class of optimization problems with certain restrictions. An open question would be to remove some of these restrictions.

**Acknowledgements.** We thank Sanjoy Dasgupta and Daniel Hsu for several pointers.

# References

[1] R. Agrawal and R. Srikant. Privacy-preserving data mining. *SIGMOD Rec.*, 29(2):439–450, 2000.

[2] B. Barak, K. Chaudhuri, C. Dwork, S. Kale, F. McSherry, and K. Talwar. Privacy, accuracy, and consistency too: a holistic solution to contingency table release. In *PODS*, pages 273–282, 2007.

[3] A. Blum, K. Ligett, and A. Roth. A learning theory approach to non-interactive database privacy. In R. E. Ladner and C. Dwork, editors, *STOC*, pages 609–618. ACM, 2008.

[4] K. Chaudhuri and N. Mishra. When random sampling preserves privacy. In C. Dwork, editor, *CRYPTO*, volume 4117 of *Lecture Notes in Computer Science*, pages 198–213. Springer, 2006.

[5] C. Dwork. Differential privacy. In M. Bugliesi, B. Preneel, V. Sassone, and I. Wegener, editors, *ICALP (2)*, volume 4052 of *Lecture Notes in Computer Science*, pages 1–12. Springer, 2006.

[6] C. Dwork, F. McSherry, K. Nissim, and A. Smith. Calibrating noise to sensitivity in private data analysis. In *Theory of Cryptography Conference*, pages 265–284, 2006.

[7] A. Evfimievski, J. Gehrke, and R. Srikant. Limiting privacy breaches in privacy preserving data mining. In *PODS*, pages 211–222, 2003.

[8] S. P. Kasiviswanathan, H. K. Lee, K. Nissim, S. Raskhodnikova, and A. Smith. What can we learn privately? In *Proc. of Foundations of Computer Science*, 2008.

[9] C. T. Kelley. *Iterative Methods for Optimization*. SIAM, 1999.

[10] A. Machanavajjhala, J. Gehrke, D. Kifer, and M. Venkitasubramaniam. l-diversity: Privacy beyond k-anonymity. In *ICDE*, page 24, 2006.

[11] F. McSherry and K. Talwar. Mechanism design via differential privacy. In *FOCS*, pages 94–103, 2007.

[12] A. Narayanan and V. Shmatikov. Robust de-anonymization of large sparse datasets. In *IEEE Symposium on Security and Privacy*, pages 111–125. IEEE Computer Society, 2008.

[13] K. Nissim, S. Raskhodnikova, and A. Smith. Smooth sensitivity and sampling in private data analysis. In D. S. Johnson and U. Feige, editors, *STOC*, pages 75–84. ACM, 2007.

[14] P. Samarati and L. Sweeney. Protecting privacy when disclosing information: k-anonymity and its enforcement through generalization and suppression. In *Proc. of the IEEE Symposium on Research in Security and Privacy*, 1998.

[15] S. Shalev-Shwartz and N. Srebro. Svm optimization: Inverse dependence on training set size. In *International Conference on Machine Learning(ICML)*, 2008.

[16] K. Sridharan, N. Srebro, and S. Shalev-Schwartz. Fast rates for regularized objectives. In *Neural Information Processing Systems*, 2008.

